# Connectionist Modeling and Parallel Architectures

**Joachim Diederich**

Neurocomputing Research Centre
School of Computing Science
Queensland University of Technology
Brisbane Q 4001 Australia

**Ah Chung Tsoi**

Department of Electrical and
Computer Engineering
University of Queensland
St Lucia, Queensland 4072, Australia

The introduction of specialized hardware platforms for connectionist modeling ("connectionist supercomputer") has created a number of research topics. Some of these issues are controversial, e.g. the efficient implementation of incremental learning techniques, the need for the dynamic reconfiguration of networks and possible programming environments for these machines.

**Joachim Diederich,** Queensland University of Technology (Brisbane), started with a brief introduction to connectionist modeling and parallel machines. Neural network modeling can be done on various levels of abstraction. On a low level of abstraction, a simulator can support the definition and simulation of "compartmental models," chemical synapses, dendritic trees etc., i.e. explicit computational models of single neurons. These models have been built by use of SPICE (UC Berkeley) and Genesis (Caltech). On a higher level of abstraction, the Rochester Connectionist Simulator (RCS; University of Rochester) and ICSIM (ICSI Berkeley) allow the definition of unit types and complex connectivity patterns. On a very high level of abstraction, simulators like tlearn (UCSD) allow the easy realization of pre-defined network architectures (feedforward networks) and learning algorithms such as backpropagation.

**Ben Gomes,** International Computer Science Institute (Berkeley) introduced the Connectionist Supercomputer 1. The CNS-1 is a multiprocessor system designed for moderate precision fixed point operations used extensively in connectionist network calculations. Custom VLSI digital processors employ an on-chip vector coprocessor unit tailored for neural network calculations and controlled by RISC scalar CPU. One processor and associated commercial DRAM comprise a node, which is connected in a mesh topology with other nodes to establish a MIMD array. One edge of the communications mesh is reserved for attaching various I/O devices, which connect via a custom network adaptor chip. The CNS-1 operates as a compute server and one I/O port is used for connecting to a host workstation.

Users with mainstream connectionist applications can use CNSim, an object-oriented, graphical high-level interface to the CNS-1 environment. Those with more complicated applications can use one of several high-level programming languages (C, C++,

Sather), and access a complete set of hand-coded assembler subroutine libraries for connectionist applications. Simulation, debugging and profiling tools will be available to aid both types of users. Additional tools are available for the systems programmer to code at a low level for maximum performance. Access to the I/O-level processor and network functions are provided, along with the evaluation tools needed to complement the process.

**Urs Muller**, Swiss Federal Institute of Technology (Zürich) introduced MUSIC: A high performance neural network simulation tool on a multiprocessor. MUSIC (Multiprocessor System with Intelligent Communication), a 64 processor system, runs backpropagation at a speed of 247 million connection updates per second using 32 bit floating-point precision. Thus the system reaches supercomputer speed (3.8 gflops peak), it still can be used as a personal desk-top computer at a researchers own disposal: The complete system consumes less than 800 Watt and fits into a 19 inch rack.

**Fin Martin**, Intel Corporation, introduced "Ni1000," an RBF processor which accepts 40,000 patterns per second. Input patterns of 256 dimensions by 5 bits are transferred from the host to the Ni1000 and compared with the chip's "memory" of 1024 stored reference patterns, in parallel. A custom 16 bit on-chip microcontroller runs at 20 MIPS and controls all the programming and algorithm functions. RBF's are considered an advancement over traditional template matching algorithms and back propagation.

**Paul Murtagh and Ah Chung Tsoi**, University of Queensland (St. Lucia) described a reconfigurable VLSI Systolic Array for artificial neural networks. After a brief review of some of the most common neural network architectures, e.g., multilayer perceptron, Hopfield net, Boltzmann machine, Ah Chung Tsoi showed that the training algorithms of these networks can be written in a unified manner. This unified training algorithm is then shown to be implementable in a systolic array fashion. The individual processor can be designed accordingly. Each processor incorporates functionality reconfiguration to allow a number of neural network models to be implemented. The architecture also incorporates reconfiguration for fault tolerance and processor arrangement. Each processor occupies very little silicon area with 16 processors being able to fit onto a 10 x 10 mm$^2$ die.

**Günther Palm and Franz Kurfess** introduced "Neural Associative Memories." Despite having processing elements which are thousands of times faster than the neurons in the brain, modern computers still cannot match quite a few processing capabilities of the brain, many of which we even consider trivial (such as recognizing faces or voices, or following a conversation). A common principle for those capabilities lies in the use of correlations between patterns in order to identify patterns which are similar. Looking at the brain as an information processing mechanism with -- probably among others -- associative processing capabilities together with the converse view of associative memories as certain types of artificial neural networks initiated a number of interesting results. These range from theoretical considerations to insights in the functioning of neurons, as well as parallel hardware implementations of neural associative memories. The talk discussed some implementation aspects and presented a few applications.

Finally, **Ernst Niebur,** California Institute of Technology (Pasadena) presented his work on biologically realistic modeling on SIMD machines (No abstract available).